# Stabilizing Value Function Approximation with the BFBP Algorithm

**Xin Wang**
Department of Computer Science
Oregon State University
Corvallis, OR, 97331
*wangxi@cs.orst.edu*

**Thomas G Dietterich**
Department of Computer Science
Oregon State University
Corvallis, OR, 97331
*tgd@cs.orst.edu*

## Abstract

We address the problem of non-convergence of online reinforcement learning algorithms (e.g., Q learning and SARSA($\lambda$)) by adopting an incremental-batch approach that separates the exploration process from the function fitting process. Our BFBP (Batch Fit to Best Paths) algorithm alternates between an exploration phase (during which trajectories are generated to try to find fragments of the optimal policy) and a function fitting phase (during which a function approximator is fit to the best known paths from start states to terminal states). An advantage of this approach is that batch value-function fitting is a global process, which allows it to address the tradeoffs in function approximation that cannot be handled by local, online algorithms. This approach was pioneered by Boyan and Moore with their GROWSUPPORT and ROUT algorithms. We show how to improve upon their work by applying a better exploration process and by enriching the function fitting procedure to incorporate Bellman error and advantage error measures into the objective function. The results show improved performance on several benchmark problems.

## 1 Introduction

Function approximation is essential for applying value-function-based reinforcement learning (RL) algorithms to solve large Markov decision problems (MDPs). However, online RL algorithms such as $SARSA(\lambda)$ have been shown experimentally to have difficulty converging when applied with function approximators. Theoretical analysis has not been able to prove convergence, even in the case of linear function approximators. (See Gordon (2001), however, for a non-divergence result.) The heart of the problem is that the approximate values of different states (e.g., $s_1$ and $s_2$) are coupled through the parameters of the function approximator. The optimal policy at state $s_1$ may require increasing a parameter, while the optimal policy at state $s_2$ may require decreasing it. As a result, algorithms based on local parameter updates tend to oscillate or even to diverge.

To avoid this problem, a more global approach is called for—an approach that

can consider $s_1$ and $s_2$ simultaneously and find a solution that works well in both states. One approach is to formulate the reinforcement learning problem as a global search through a space of parameterized policies as in the policy gradient algorithms (Williams, 1992; Sutton, McAllester, Singh, & Mansour, 2000; Konda & Tsitsiklis, 2000; Baxter & Bartlett, 2000). This avoids the oscillation problem, but the resulting algorithms are slow and only converge to local optima.

We pursue an alternative approach that formulates the function approximation problem as a global supervised learning problem. This approach, pioneered by Boyan and Moore in their GROWSUPPORT (1995) and ROUT (1996) algorithms, separates the reinforcement learning problem into two subproblems: the exploration problem (finding a good partial value function) and the representation problem (representing and generalizing that value function). These algorithms alternate between two phases. During the exploration phase, a *support set* of points is constructed whose optimal values are known within some tolerance. In the function fitting phase, a function approximator $\hat{V}$ is fit to the support set.

In this paper, we describe two ways of improving upon GROWSUPPORT and ROUT. First, we replace the support set with the set of states that lie along the best paths found during exploration. Second, we employ a combined error function that includes terms for the supervised error, the Bellman error, and the advantage error (Baird, 1995) into the function fitting process. The resulting BFBP (Batch Fit to Best Paths) method gives significantly better performance on resource-constrained scheduling problems as well as on the mountain car toy benchmark problem.

## 2 GrowSupport, ROUT, and BFBP

Consider a deterministic, episodic MDP. Let $s' = a(s)$ denote the state $s'$ that results from performing $a$ in $s$ and $r(a, s)$ denote the one-step reward. Both GROWSUPPORT and ROUT build a support set $S = \{(s_i, V(s_i))\}$ of states whose optimal values $V(s)$ are known with reasonable accuracy. Both algorithms initialize $S$ with a set of terminal states (with $V(s) = 0$). In each iteration, a function approximator $\hat{V}$ is fit to $S$ to minimize $\sum_i [V(s_i) - \hat{V}(s_i)]^2$. Then, an exploration process attempts to identify new points to include in $S$.

In GROWSUPPORT, a sample of points $X$ is initially drawn from the state space. In each iteration, after fitting $\hat{V}$, GROWSUPPORT computes a new estimate $V(s)$ for each state $s \in X$ according to $V(s) = \max_a r(s, a) + V(a(s))$, where $V(a(s))$ is computed by executing the greedy policy with respect to $\hat{V}$ starting in $a(s)$. If $V(a(s))$ is within $\epsilon$ of $\hat{V}(a(s))$, for all actions $a$, then $(s, V(s))$ is added to $S$.

ROUT employs a different procedure suitable for stochastic MDPs. Let $P(s'|s, a)$ be the probability that action $a$ in state $s$ results in state $s'$ and $R(s'|s, a)$ be the expected one-step reward. During the exploration phase, ROUT generates a trajectory from the start state to a terminal state and then searches for a state $s$ along that trajectory such that (i) $\hat{V}(s)$ is *not* a good approximation to the backed-up value $V(s) = \max_a \sum_{s'} P(s'|s, a)[R(s'|s, a) + \hat{V}(s')]$, and (ii) for every state $\bar{s}$ along a set of rollout trajectories starting at $s'$, $\hat{V}(\bar{s})$ is within $\epsilon$ of the backed-up value $\max_a \sum_{s'} P(s'|\bar{s}, a)[R(s'|\bar{s}, a) + \hat{V}(s')]$. If such a state is found, then $(s, V(s))$ is added to $S$.

Both GROWSUPPORT and ROUT rely on the function approximator to generalize well at the boundaries of the support set. A new state $s$ can only be added to $S$ if $\hat{V}$ has generalized to all of $s$'s successor states. If this occurs consistently,

then eventually the support set will expand to include all of the starting states of the MDP, at which point a satisfactory policy has been found. However, if this "boundary generalization" does not occur, then no new points will be added to $S$, and both GROWSUPPORT and ROUT terminate without a solution. Unfortunately, most regression methods have high bias and variance near the boundaries of their training data, so failures of boundary generalization are common.

These observations led us to develop the BFBP algorithm. In BFBP, the exploration process maintains a data structure $S$ that stores the best known path from the start state to a terminal state and a "tree" of one-step departures from this best path (i.e., states that can be reached by executing an action in some state on the best path). At each state $s_i \in S$, the data structure stores the action $a_i^*$ executed in that state (to reach the next state in the path), the one-step reward $r_i$, and the estimated value $V(s_i)$. $S$ also stores each action $a_-$ that causes a departure from the best path along with the resulting state $s_-$, reward $r_-$ and estimated value $V(s_-)$. We will denote by $B$ the subset of $S$ that constitutes the best path. The estimated values $V$ are computed as follows. For states $s_i \in B$, $V(s_i)$ is computed by summing the immediate rewards $r_j$ for all steps $j \geq i$ along $B$. For the one-step departure states $s_-$, $V(s_-)$ is computed from an exploration trial in which the greedy policy was followed starting in state $s_-$.

Initially, $S$ is empty, so a random trajectory is generated from the start state $s_0$ to a terminal state, and it becomes the initial best known path. In subsequent iterations, a state $s_i \in B$ is chosen at random, and an action $a' \neq a_i^*$ is chosen and executed to produce state $s'$ and reward $r'$. Then the greedy policy (with respect to the current $\hat{V}$) is executed until a terminal state is reached. The rewards along this new path are summed to produce $V(s')$. If $V(s') + r' > V(s_i)$, then the best path is revised as follows. The new best action in state $s_i$ becomes $a'$ with estimated value $V(s') + r'$. This improved value is then propagated backwards to update the $V$ estimates for all ancestor states in $B$. The old best action $a_i^*$ in state $s_i$ becomes an inferior action $a_-$ with result state $s_-$. Finally all descendants of $s_-$ along the old best path are deleted. This method of investigating one-step departures from the best path is inspired by Harvey and Ginsberg's (1995) limited discrepancy search (LDS) algorithm. In each exploration phase, $K$ one-step departure paths are explored.

After the exploration phase, the value function approximation $\hat{V}$ is recomputed with the goal of minimizing a combined error function:

$$J(\hat{V}) = \lambda_s \sum_{s \in S} \left( \hat{V}(s) - V(s) \right)^2 + \lambda_b \sum_{s \in B} \left( \hat{V}(s) - [r(s, a^*) + \hat{V}(a^*(s))] \right)^2 +$$
$$\lambda_a \sum_{s \in B} \sum_{a_- \neq a^*} \left( [r(s, a_-) + \hat{V}(a_-(s))] - [r(s, a^*) + \hat{V}(a^*(s))] \right)_+^2 .$$

The three terms of this objective function are referred to as the supervised, Bellman, and advantage terms. Their relative importance is controlled by the coefficients $\lambda_s$, $\lambda_b$, and $\lambda_a$. The supervised term is the usual squared error between the $V(s)$ values stored in $S$ and the fitted values $\hat{V}(s)$. The Bellman term is the squared error between the fitted value and the backed-up value of the next state on the best path. And the advantage term penalizes any case where the backed-up value of an inferior action $a_-$ is larger than the backed-up value of the best action $a^*$. The notation $(u)_+ = u$ if $u \geq 0$ and 0 otherwise.

**Theorem 1** *Let $M$ be a deterministic MDP such that (a) there are only a finite number of starting states, (b) there are only a finite set of actions executable in each state, and (c) all policies reach a terminal state. Then BFBP applied to $M$ converges.*

**Proof:** The LDS exploration process is monotonic, since the data structure $S$ is only updated if a new best path is found. The conditions of the theorem imply that there are only a finite number of possible paths that can be explored from the starting states to the terminal states. Hence, the data structure $S$ will eventually converge. Consequently, the value function $\hat{V}$ fit to $S$ will also converge. **Q.E.D.**

The theorem requires that the MDP contain no cycles. There are cycles in our job-shop scheduling problems, but we eliminate them by remembering all states visited along the current trajectory and barring any action that would return to a previously visited state. Note also that the theorem applies to MDPs with continuous state spaces provided the action space and the start states are finite.

Unfortunately, BFBP does not necessarily converge to an optimal policy. This is because LDS exploration can get stuck in a local optimum such that all one step departures using the $\hat{V}$-greedy policy produce trajectories that do not improve over the current best path. Hence, although BFBP resembles policy iteration, it does not have the same optimality guarantees, because policy iteration evaluates the current greedy policy in *all* states in the state space.

Theoretically, we could prove convergence to the optimal policy under modified conditions. If we replace LDS exploration with $\epsilon$-greedy exploration, then exploration will converge to the optimal paths with probability 1. When trained on those paths, if the function approximator fits a sufficiently accurate $\hat{V}$, then BFBS will converge optimally. In our experiments, however, we have found that $\epsilon$-greedy gives no improvement over LDS, whereas LDS exploration provides more complete coverage of one-step departures from the current best path, and these are used in $J(\hat{V})$.

## 3   Experimental Evaluation

We have studied five domains: Grid World and Puddle World (Boyan & Moore, 1995), Mountain Car (Sutton, 1996), and resource-constrained scheduling problems ART-1 and ART-2 (Zhang & Dietterich, 1995). For the first three domains, following Boyan and Moore, we compare BFBP with GROWSUPPORT. For the final domain, it is difficult to draw a sample of states $X$ from the state space to initialize GROWSUPPORT. Hence, we compare against ROUT instead. As mentioned above, we detected and removed cycles from the scheduling domain (since ROUT requires this). We retained the cycles in the first three problems. On mountain car, we also applied $SARSA(\lambda)$ with the CMAC function approximator developed by Sutton (1996).

We experimented with two function approximators: regression trees (RT) and locally-weighted linear regression (LWLR). Our regression trees employ linear separating planes at the internal nodes and linear surfaces at the leaf nodes. The trees are grown top-down in the usual fashion. To determine the splitting plane at a node, we choose a state $s_i$ at random from $S$, choose one of its inferior children $s_-$, and construct the plane that is the perpendicular bisector of these two points. The splitting plane is evaluated by fitting the resulting child nodes to the data (as leaf nodes) and computing the value of $J(\hat{V})$. A number $C$ of parent-child pairs $(s_i, s_-)$ are generated and evaluated, and the best one is retained to be the splitting plane. This process is then repeated recursively until a node contains fewer than $M$ data points. The linear surfaces at the leaves are trained by gradient descent to minimize $J(\hat{V})$. The gradient descent terminates after 100 steps or earlier if $J$ becomes very small. In our experiments, we tried all combinations of the following parameters and report the best results: (a) 11 learning rates (from 0.00001 to 0.1), (b) $M = 1$,

Table 1: Comparison of results on three toy domains.

| Problem Domain | Algorithms | Optimal Policy? | Best Policy Length |
|---|---|---|---|
| Grid World | GROWSUPPORT | Yes | **39** |
| | BFBP | Yes | **39** |
| Puddle World | GROWSUPPORT | Yes | **39** |
| | BFBP | Yes | **39** |
| Mountain Car | SARSA($\lambda$) | No | 103 |
| | GROWSUPPORT | No | 93 |
| | BFBP | Yes | 88 |

Table 2: Results of ROUT and BFBP on scheduling problem ART-1-TRN00

| Performance | ROUT (RT) | ROUT (LWLR) | BFBP (RT) |
|---|---|---|---|
| Best policy explored | 1.75 | 1.55 | 1.50 |
| Best final learned policy | 1.8625 | 1.8125 | 1.55 |

10, 20, 50, 100, 1000, (c) $C = 5, 10, 20, 50, 100$, and (d) $K = 50, 100, 150, 200$.

For locally-weighted linear regression, we replicated the methods of Boyan and Moore. To compute $\hat{V}(s)$, a linear regression is performed using all points $s_i \in S$ weighted by their distance to $s$ according to the kernel $\exp -(\|s_i - s\|^2/\sigma^2)$. We experimented with all combinations of the following parameters and report the best results: (a) 29 values (from 0.01 to 1000.0) of the tolerance $\epsilon$ that controls the addition of new points to $S$, and (b) 39 values (from 0.01 to 1000.0) for $\sigma$.

We execute ROUT and GROWSUPPORT to termination. We execute BFBP for 100 iterations, but it converges much earlier: 36 iterations for the grid world, 3 for puddle world, 10 for mountain car, and 5 for the job-shop scheduling problems.

Table 1 compares the results of the algorithms on the toy domains with parameters for each method tuned to give the best results and with $\lambda_s = 1$ and $\lambda_b = \lambda_a = 0$. In all cases, BFBP matches or beats the other methods. In Mountain Car, in particular, we were pleased that BFBP discovered the optimal policy very quickly. Table 2 compares the results of ROUT and BFBP on job-shop scheduling problem TRN00 from problem set ART-1 (again with $\lambda_s = 1$ and $\lambda_b = \lambda_a = 0$). For ROUT, results with both LWLR and RT are shown. LWLR gives better results for ROUT. We conjecture that this is because ROUT needs a value function approximator that is conservative near the boundary of the training data, whereas BFBP does not. We report both the best policy found during the iterations and the final policy at convergence. Figure 1 plots the results for ROUT (LWLR) against BFBP (RT) for eight additional scheduling problems from ART-1. The figure of merit is RDF, which is a normalized measure of schedule length (small values are preferred). BFBP's learned policy out-performs ROUT's in every case.

The experiments above all employed only the supervised term in the error function $J$. These experiments demonstrate that LDS exploration gives better training sets than the support set methods of GROWSUPPORT and ROUT. Now we turn to the question of whether the Bellman and advantage terms can provide improved results. For the grid world and puddle world tasks, the supervised term already gives optimal performance, so we focus on the mountain car and job-shop scheduling problems.

Table 3 summarizes the results for BFBP on the mountain car problem. All parameter settings, except for the last, succeed in finding the optimal policy. To get

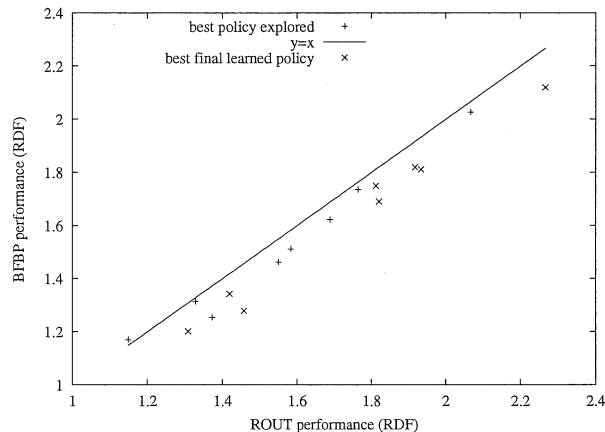

Figure 1: Performance of Rout vs. BFBP over 8 job shop scheduling problems

Table 3: Fraction of parameter settings that give optimal performance for BFBP on the mountain car problem

| $\lambda_s$ | $\lambda_b$ | $\lambda_a$ | # settings | $\lambda_s$ | $\lambda_b$ | $\lambda_a$ | # settings |
|---|---|---|---|---|---|---|---|
| 0.0 | 0.0 | 1.0 | 2/1311 | 0.0 | 1.0 | 0.0 | 1/1297 |
| 1.0 | 0.0 | 0.0 | 52/1280 | 1.0 | 0.0 | 10.0 | **184/1291** |
| 1.0 | 10.0 | 0.0 | 163/1295 | 1.0 | 0.0 | 100.0 | 133/1286 |
| 1.0 | 100.0 | 0.0 | 4/939 | 1.0 | 1000.0 | 0.0 | 0/1299 |

a sense of the robustness of the method, we also report the fraction of parameter settings that gave the optimal policy. The number of parameter settings tested (the denominator) should be the same for all combinations of $\lambda$ values. Nonetheless, for reasons unrelated to the parameter settings, some combinations failed to be executed by our distributed process scheduler. The best settings combine $\lambda_s = 1$ with either $\lambda_b = 10$ or $\lambda_a = 10$. However, if we employ either the Bellman or the advantage term alone, the results are poor. Hence, it appears that the supervised term is very important for good performance, but that the advantage and Bellman terms can improve performance substantially and reduce the sensitivity of BFBP to the settings of the other parameters.

Table 4 shows the performance of BFBP on ART-1-TRN00. The best performance (at convergence) is obtained with $\lambda_s = \lambda_a = 1$ and $\lambda_b = 0$. As with mountain car, these experiments show that the supervised term is the most important, but that it gives even better results when combined with the advantage term.

All of the above experiments compare performance on single problems. We also tested the ability of BFBP to generalize to similar problems following the formulation of (Zhang & Dietterich, 1995). Figure 2 compares the performance of neural networks and regression trees as function approximators for BFBP. Both were trained on job shop scheduling problem set ART-2. Twenty of the problems in ART-2 were used for training, 20 for cross-validation, and 50 for testing. Eleven different values for $\lambda_s$, $\lambda_b$, $\lambda_a$ and eight different values for the learning rate were tried, with the best values selected according to the cross-validation set. Figure 2 shows that BFBP is significantly better than the baseline performance (with RDF

Table 4: Performance of BFBP on ART-1-TRN00 for different settings of the $\lambda$ parameters. The "perform" column gives the best RDF in any iteration and the RDF at convergence.

| $\lambda_s$ | $\lambda_b$ | $\lambda_a$ | perform. | $\lambda_s$ | $\lambda_b$ | $\lambda_a$ | perform. | $\lambda_s$ | $\lambda_b$ | $\lambda_a$ | perform. |
|---|---|---|---|---|---|---|---|---|---|---|---|
| 0 | 0 | 1 | 1.50/1.75 | 0 | 1 | 0 | 1.50/1.775 | 1 | 0 | 0 | 1.50/1.55 |
| 0 | 1 | 1 | 1.50/1.775 | 0 | 1 | 10 | 1.50/1.825 | 0 | 1 | 100 | 1.50/1.65 |
| 0 | 10 | 1 | 1.50/1.775 | 0 | 100 | 1 | 1.50/1.738 | 1 | 1 | 0 | 1.50/1.563 |
| 1 | 0 | 1 | **1.50/1.488** | 1 | 0 | 10 | 1.463/1.525 | 1 | 0 | 100 | 1.50/1.588 |
| 1 | 1 | 1 | 1.525/1.55 | 1 | 1 | 10 | 1.50/1.588 | 1 | 1 | 100 | 1.50/1.675 |

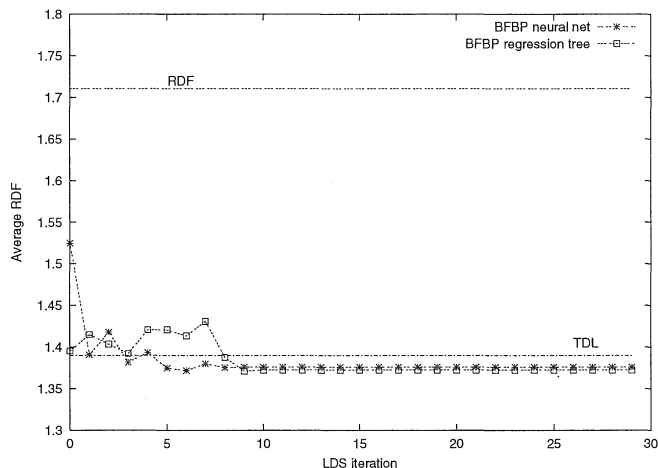

Figure 2: BFBP on ART-2 using neural nets and regression trees. "RDF" is a hand-coded heuristic, "TDL" is Zhang's $TD(\lambda)$ neural network.

as a search heuristic) and that its performance is comparable to TD($\lambda$) with neural networks (Zhang & Dietterich, 1995). Figure 3 shows that for ART-2, using parent/inferior-child pair splits gives better results than using axis-parallel splits.

## 4   Conclusions

This paper has shown that the exploration strategies underlying GROWSUPPORT and ROUT can be improved by simply remembering and training on the best paths found between start and terminal states. Furthermore, the paper proved that the BFBP method converges for arbitrary function approximators, which is a result that has not yet been demonstrated for online methods such as SARSA($\lambda$). In addition, we have shown that the performance of our BFBP algorithm can be further improved (and made more robust) by incorporating a penalty for violations of the Bellman equation or a penalty for preferring inferior actions (an advantage error).

Taken together, these results show that incremental-batch value function approximation can be a reliable, convergent method for solving deterministic reinforcement learning problems. The key to the success of the method is the ability to separate the exploration process from the function approximation process and to make the exploration process convergent. This insight should also be applicable to stochastic episodic MDPs.

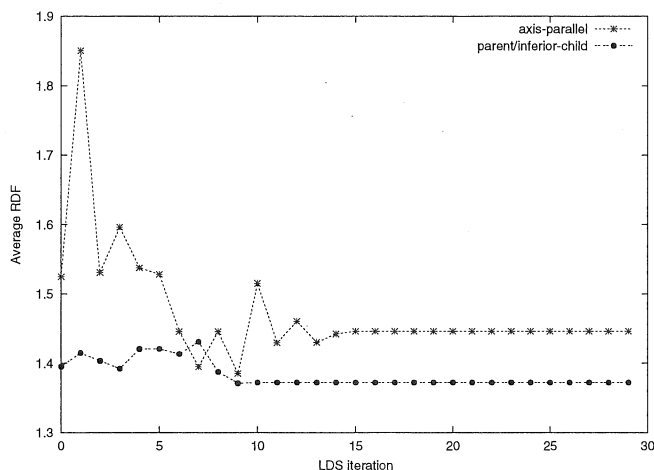

Figure 3: Axis parallel splits versus parent/inferior-child pair splits on ART-2

## Acknowledgments

The authors gratefully acknowledge the support of AFOSR under contract F49620-98-1-0375, and the NSF under grants IRI-9626584, IIS-0083292, ITR-5710001197, and EIA-9818414. We thank Valentina Zubek for her careful reading of the paper.

## References

Baird, L. C. (1995). Residual algorithms: Reinforcement learning with function approximation. In *ICML-95*, 30–37, San Francisco, CA. Morgan Kaufmann.

Baxter, J., & Bartlett, P. L. (2000). Reinforcement learning in POMDP's via direct gradient ascent. In *ICML-2000*, 41–48. Morgan Kaufmann, San Francisco, CA.

Boyan, J. A., & Moore, A. W. (1995). Generalization in reinforcement learning: Safely approximating the value function. In *NIPS-7*, 369–376. The MIT Press, Cambridge.

Boyan, J. A., & Moore, A. W. (1996). Learning evaluation functions for large acyclic domains. In *ICML-96*, 63–70. Morgan Kaufmann, San Francisco, CA.

Gordon, G. J. (2001). Reinforcement learning with function approximation converge to a region. In *NIPS-13*, 1040–1046. The MIT Press.

Harvey, W. D., & Ginsberg, L. P. (1995). Limited discrepancy search. In *IJCAI-95*, 825–830. Morgan Kaufmann.

Konda, V. R., & Tsitsiklis, J. N. (2000). Policy gradient methods for reinforcement learning with function approximation. In *NIPS-12*, 1008–1014 Cambridge, MA. MIT Press.

Moll, R., Barto, A. G., Perkins, T. J., & Sutton, R. S. (1999). Learning instance-independent value functions to enhance local search. In *NIPS-11*, 1017–1023.

Sutton, R. S., McAllester, D., Singh, S., & Mansour, Y. (2000). Policy gradient methods for reinforcement learning with function approximation. In *NIPS-12*, 1057–1063.

Sutton, R. S. (1996). Generalization in reinforcement learning: Successful examples using sparse coarse coding. In *NIPS-8*, 1038–1044. The MIT Press, Cambridge.

Williams, R. J. (1992). Simple statistical gradient-following algorithms for connectionist reinforcement learning. *Machine Learning, 8*, 229.

Zhang, W., & Dieterich, T. G. (1995). A reinforcement learning approach to job-shop scheduling. In *IJCAI-95*, 1114–1120. Morgan Kaufmann, San Francisco, CA.
